# A Comparison of Projection Pursuit and Neural Network Regression Modeling

**Jenq-Neng Hwang, Hang Li,**
Information Processing Laboratory
Dept. of Elect. Engr., FT-10
University of Washington
Seattle WA 98195

**Martin Maechler, R. Douglas Martin, Jim Schimert**
Department of Statistics
Mail Stop: GN-22
University of Washington
Seattle, WA 98195

## Abstract

Two projection based feedforward network learning methods for model-free regression problems are studied and compared in this paper: one is the popular *back-propagation* learning (BPL); the other is the *projection pursuit* learning (PPL). Unlike the totally parametric BPL method, the PPL non-parametrically estimates unknown nonlinear functions sequentially (neuron-by-neuron and layer-by-layer) at each iteration while jointly estimating the interconnection weights. In terms of learning efficiency, both methods have comparable training speed when based on a Gauss-Newton optimization algorithm while the PPL is more parsimonious. In terms of learning robustness toward noise outliers, the BPL is more sensitive to the outliers.

## 1   INTRODUCTION

The back-propagation learning (BPL) networks have been used extensively for essentially two distinct problem types, namely model-free *regression* and *classification*, which have no *a priori* assumption about the unknown functions to be identified other than imposes a certain degree of smoothness. The projection pursuit learning (PPL) networks have also been proposed for both types of problems (Friedman85 [3]), but to date there appears to have been much less actual use of PPLs for both regression and classification than of BPLs. In this paper, we shall concentrate on regression modeling applications of BPLs and PPLs since the regression setting is one in which some fairly deep theory is available for PPLs in the case of low-dimensional regression (Donoho89 [2], Jones87 [6]).

A multivariate model-free regression problem can be stated as follows: given $n$ pairs of vector observations, $(\mathbf{y}_l, \mathbf{x}_l) = (y_{l1}, \cdots, y_{lq}; x_{l1}, \cdots, x_{lp})$, which have been generated from unknown models

$$y_{li} = g_i(\mathbf{x}_l) + \epsilon_{li}, \quad l = 1, 2, \cdots, n; \quad i = 1, 2, \cdots, q \qquad (1)$$

where $\{\mathbf{y}_l\}$ are called the multivariable "response" vector and $\{\mathbf{x}_l\}$ are called the "independent variables" or the "carriers". The $\{g_i\}$ are unknown smooth non-parametric (model-free) functions from $p$-dimensional Euclidean space to the real line, i.e., $g_i : R^p \longrightarrow R, \forall i$. The $\{\epsilon_{li}\}$ are random variables with zero mean, $E[\epsilon_{li}] = 0$, and independent of $\{\mathbf{x}_l\}$. Often the $\{\epsilon_{li}\}$ are assumed to be independent and identically distributed (*iid*) as well.

The goal of regression is to generate the estimators, $\hat{g}_1, \hat{g}_2, \cdots, \hat{g}_q$, to best approximate the unknown functions, $g_1, g_2, \cdots, g_q$, so that they can be used for prediction of a new $\mathbf{y}$ given a new $\mathbf{x}$: $\hat{y}_i = \hat{g}_i(\mathbf{x}), \forall i$.

## 2 A TWO-LAYER PERCEPTRON AND BACK-PROPAGATION LEARNING

Several recent results have shown that a two-layer (one hidden layer) perceptron with sigmoidal nodes can in principle represent any Borel-measurable function to any desired accuracy, assuming "enough" hidden neurons are used. This, along with the fact that theoretical results are known for the PPL in the analogous two-layer case, justifies focusing on the two-layer perceptron for our studies here.

### 2.1 MATHEMATICAL FORMULATION

A two-layer perceptron can be mathematically formulated as follows:

$$
\begin{aligned}
u_k &= \sum_{j=1}^{p} w_{kj} x_j - \theta_k = \mathbf{w}_k^T \mathbf{x} - \theta_k, \quad k = 1, 2, \cdots, m \\
y_i &= \sum_{k=1}^{m} \beta_{ik} f_k(u_k) = \sum_{k=1}^{m} \beta_{ik} \sigma(u_k), \quad i = 1, 2, \cdots, q
\end{aligned}
\qquad (2)
$$

where $u_k$ denotes the weighted sum input of the $k^{th}$ neuron in the hidden layer; $\theta_k$ denotes the bias of the $k^{th}$ neuron in the hidden layer; $w_{kj}$ denotes the input-layer weight linked between the $k^{th}$ hidden neuron and the $j^{th}$ neuron of the input

layer (or $j^{th}$ element of the input vector $\mathbf{x}$); $\beta_{ik}$ denotes the output-layer weight linked between the $i^{th}$ output neuron and the $k^{th}$ hidden neuron; $f_k$ is the nonlinear activation function, which is usually assumed to be a fixed monotonically increasing (logistic) sigmoidal function, $\sigma(u) = 1/(1 + e^{-u})$.

The above formulation defines quite explicitly the parametric representation of functions which are being used to approximate $\{g_i(\mathbf{x}), \ i = 1, 2, \cdots, q\}$. A simple reparametrization allows us to write $\hat{g}_i(\mathbf{x})$ in the form:

$$\hat{g}_i(\mathbf{x}) = \sum_{k=1}^{m} \beta_{ik} \sigma\left(\frac{\alpha_k^T \mathbf{x} - \mu_k}{s_k}\right) \tag{3}$$

where $\alpha_k$ is a unit length version of weight vector $\mathbf{w}_k$. This formulation reveals how $\{\hat{g}_i\}$ are built up as a linear combination of sigmoids evaluated at translates (by $\mu_k$) and scaled (by $s_k$) projection of $\mathbf{x}$ onto the unit length vector $\alpha_k$.

## 2.2 BACK-PROPAGATION LEARNING AND ITS VARIATIONS

Historically, the training of a multilayer perceptron uses back-propagation learning (BPL). There are two common types of BPL: the *batch* one and the *sequential* one. The batch BPL updates the weights after the presentation of the complete set of training data. Hence, a training iteration incorporates one sweep through all the training patterns. On the other hand, the sequential BPL adjusts the network parameters as training patterns are presented, rather than after a complete pass through the training set. The sequential approach is a form of Robbins-Monro *Stochastic Approximation*.

While the two-layer perceptron provides a very powerful nonparametric modeling capability, the BPL training can be slow and inefficient since only the first derivative (or gradient) information about the training error is utilized. To speed up the training process, several *second-order* optimization algorithms, which take advantage of second derivative (or Hessian matrix) information, have been proposed for training perceptrons (Hwang90 [4]). For example, the Gauss-Newton method is also used in the PPL (Friedman85 [3]).

The fixed nonlinear nodal (sigmoidal) function is a monotone nondecreasing differentiable function with very simple first derivative form, and possesses nice properties for numerical computation. However, it does not interpolate/extrapolate efficiently in a wide variety of regression applications. Several attempts have been proposed to improve the choice of nonlinear nodal functions; e.g., the Gaussian or bell-shaped function, the locally tuned radial basis functions, and semi-parametric (non-fixed nodal function) nonlinear functions used in PPLs and hidden Markov models.

## 2.3 RELATIONSHIP TO KERNEL APPROXIMATION AND DATA SMOOTHING

It is instructive to compare the two-layer perceptron approximation in Eq. (3) with the well-known kernel method for regression. A *kernel* $K(\cdot)$ is a non-negative symmetric function which integrates to unity. Most kernels are also unimodal, with

mode at the origin, $K(t_1) \geq K(t_2)$, $0 \leq t_1 < t_2$. A kernel estimate of $g_i(\mathbf{x})$ has the form

$$\hat{g}_{K,i}(\mathbf{x}) = \sum_{l=1}^{n} y_{li} \frac{1}{h^q} K(\frac{\|\mathbf{x} - \mathbf{x}_l\|}{h^q}), \tag{4}$$

where $h$ is a bandwidth parameter and $q$ is the dimension of $\mathbf{y}_l$ vector. Typically a good value of $h$ will be chosen by a data-based cross-validation method. Consider for a moment the special case of the kernel approximator and the two-layer perceptron in Eq. (3) respectively, with *scalar* $y_l$ and $x_l$, i.e., with $p = q = 1$ (hence unit length interconnection weight $\alpha = 1$ by definition):

$$\hat{g}_K(x) = \sum_{l=1}^{n} y_l \frac{1}{h} K(\frac{\|x - x_l\|}{h}) = \sum_{l=1}^{n} y_l \frac{1}{h} K(\frac{x - x_l}{h}), \tag{5}$$

$$\hat{g}(x) = \sum_{k=1}^{m} \beta_k \sigma(\frac{x - \mu_k}{s_k}) \tag{6}$$

This reveals some important connections between the two approaches.

Suppose that for $\hat{g}(x)$, we set $\sigma = K$, i.e., $\sigma$ is a kernel and in fact identical to the kernel $K$, and that $\beta_k, \mu_k, s_k \equiv s$ have been chosen (trained), say by BPL. That is, all $\{s_k\}$ are constrained to a single unknown parameter value $s$. In general, $m \leq n$, or even $m$ is a modest fraction of $n$ when the unknown function $g(x)$ is reasonably smooth. Furthermore, suppose that $h$ has been chosen by cross validation. Then one can expect $\hat{g}_K(x) \approx \hat{g}_\sigma(x)$, particularly in the event that the $\{\mu_k\}$ are close to the observed values $\{x_l\}$ and $x$ is close to a specific $\mu_k$ value (relative to $h$). However, in this case where we force $s_k \equiv s$, one might expect $\hat{g}_K(x)$ to be a somewhat better estimate overall than $\hat{g}_\sigma(x)$, since the former is more local in character.

On the other hand, when one removes the restriction $s_k \equiv s$, then BPL leads to a local bandwidth selection, and in this case one may expect $\hat{g}_\sigma(x)$ to provide better approximation than $\hat{g}_K(x)$ when the function $g(x)$ has considerably varying curvature, $g''(x)$, and/or considerably varying error variance for the noise $\epsilon_{li}$ in Eq. (1). The reason is that a fixed bandwidth kernel estimate can not cope as well with changing curvature and/or noise variance as can a good smoothing method which uses a good local bandwidth selection method. A small caveat is in order: if $m$ is fairly large, the estimation of a separate bandwidth for each kernel location, $\mu_k$, may cause some increased variability in $\hat{g}_\sigma(x)$ by virtue of using many more parameters than are needed to adequately represent a nearly optimal local bandwidth selection method. Typically a nearly optimal local bandwidth function will have some degree of smoothness, which reflects smoothly varying curvature and/or noise variance, and a good local bandwidth selection method should reflect the smoothness constraints. This is the case in the high-quality "supersmoother", designed for applications like the PPL (to be discussed), which uses cross-validation to select bandwidth locally (Friedman85 [3]), and combines this feature with considerable speed.

The above arguments are probably equally valid without the restriction $\sigma = K$, because two sigmoids of opposite signs (via choice of two $\{\beta_k\}$) that are appropriately

shifted, will approximate a kernel up to a scaling to enforce unity area. However, there is a novel aspect: one can have a separate local bandwidth for each half of the kernel, thereby using an asymmetric kernel, which might improve the approximation capabilities relative to symmetric kernels with a single local bandwidth in some situations.

In the multivariate case, the curse of dimensionality will often render useless the kernel approximator $\hat{g}_{K,i}(\mathbf{x})$ given by Eq. (4). Instead one might consider using a projection pursuit kernel (PPK) approximator:

$$\hat{g}_{PPK,i}(\mathbf{x}) = \sum_{l=1}^{n} \sum_{k=1}^{m} y_{li} \frac{1}{h_k} K\left(\frac{\alpha_k^T \mathbf{x} - \alpha_k^T \mathbf{x}_l}{h_k}\right) \tag{7}$$

where a different bandwidth $h_k$ is used for each direction $\alpha_k$. In this case, the similarities and differences between the PPK estimate and the BPL estimate $\hat{g}_{\sigma,i}(\mathbf{x})$ become evident.

The main difference between the two methods is that PPK performs explicit smoothing in each direction $\alpha_k$ using a kernel smoother, whereas BPL does implicit smoothing with both $\beta_k$ (replacing $y_{li}/h_k$) and $\mu_k$ (replacing $\alpha_k^T \mathbf{x}_l$) being determined by nonlinear least squares optimization. In both PPK and BPL, the $\alpha_k$ and $h_k$ are determined by nonlinear optimization (cross-validation choices of bandwidth parameters are inherently nonlinear optimization problems) (Friedman85 [3]).

# 3  PROJECTION PURSUIT LEARNING NETWORKS

The projection pursuit learning (PPL) is a statistical procedure proposed for multivariate data analysis using a two-layer network given in Eq. (2). This procedure derives its name from the fact that it interprets high dimensional data through well-chosen lower-dimensional projections. The "pursuit" part of the name refers to optimization with respect to the projection directions.

## 3.1  COMPARATIVE STRUCTURES OF PPL AND BPL

Similar to a BPL perceptron, a PPL network forms projections of the data in directions determined from the interconnection weights. However, unlike a BPL perceptron, which employs a fixed set of nonlinear (sigmoidal) functions, a PPL non-parametrically estimates the nonlinear nodal functions based on nonlinear optimization approach which involves use of a one-dimensional data-smoother (e.g., a least squares estimator followed by a variable window span data averaging mechanism) (Friedman85 [3]). Therefore, it is important to note that a PPL network is a semi-parametric learning network, which consists of both parametrically and non-parametrically estimated elements. This is in contrast to a BPL perceptron, which is a completely parametric model.

## 3.2  LEARNING STRATEGIES OF PPL

In comparison with a batch BPL, which employs either 1st-order gradient descent or 2nd-order Newton-like methods to estimate the weights of all layers *simultaneously*

after all the training patterns are presented, a PPL learns neuron-by-neuron and layer-by-layer *cyclically* after all the training patterns are presented. Specifically, it applies linear least squares to estimate the output-layer weights, a one-dimensional data smoother to estimate the nonlinear nodal functions of each hidden neuron, and the Gauss-Newton nonlinear least squares method to estimate the input-layer weights.

The PPL procedure uses the batch learning technique to iteratively minimize the mean squared error, $E$, over all the training data. All the parameters to be estimated are hierarchically divided into $m$ groups (each associated with one hidden neuron), and each group, say the $k^{th}$ group, is further divided into three subgroups: the output-layer weights, $\{\beta_{ik}, \ i = 1, \cdots, q\}$, connected to the $k^{th}$ hidden neuron; the nonlinear function, $f_k(u)$, of the $k^{th}$ hidden neuron; and the input-layer weights, $\{w_{kj}, \ j = 1, \cdots, p\}$, connected to the $k^{th}$ hidden neuron. The PPL starts from updating the parameters associated with the first hidden neuron (group) by updating each subgroup, $\{\beta_{i1}\}$, $f_1(u)$, and $\{w_{1j}\}$ consecutively (layer-by-layer) to minimize the mean squared error $E$. It then updates the parameters associated with the second hidden neuron by consecutively updating $\{\beta_{i2}\}$, $f_2(u)$, and $\{w_{2j}\}$. A complete updating pass ends at the updating of the parameters associated with the $m^{th}$ (the last) hidden neuron by consecutively updating $\{\beta_{im}\}$, $f_m(u)$, and $\{w_{mj}\}$. Repeated updating passes are made over all the groups until convergence (i.e., in our studies of Section 4, we use the stopping criterion that $\frac{|E^{(new)} - E^{(old)}|}{E^{(old)}}$ be smaller than a prespecified small constant, $\xi = 0.005$).

# 4    LEARNING EFFICIENCY IN BPL AND PPL

Having discussed the "parametric" BPL and the "semi-parametric" PPL from structural, computational, and theoretical viewpoints, we have also made a more practical comparison of learning efficiency via a simulation study. For simplicity of comparison, we confine the simulations to the two-dimensional univariate case, i.e., $p = 2$, $q = 1$. This is an important situation in practice, because the models can be visualized graphically as functions $y = g(x_1, x_2)$.

## 4.1    PROTOCOLS OF THE SIMULATIONS

**Nonlinear Functions:**    There are five nonlinear functions $g^{(j)} : [0, 1]^2 \rightarrow \mathbb{R}$ investigated (Maechler90 [7]), which are *scaled* such that the standard deviation is 1 (for a large regular grid of 2500 points on $[0, 1]^2$), and *translated* to make the range nonnegative.

**Training and Test Data:**    Two independent variables (carriers) $(x_{l1}, \ x_{l2})$ were generated from the uniform distribution $U([0, 1]^2)$, i.e., the abscissa values $\{(x_{l1}, \ x_{l2})\}$ were generated as uniform random variates on $[0, 1]$ and independent from each other. We generated 225 pairs $\{(x_{l1}, \ x_{l2})\}$ of abscissa values, and used this same set for experiments of all five different functions, thus eliminating an unnecessary extra random component of the simulation. In addition to one set of noiseless training data, another set of noisy training data was also generated by adding *iid* Gaussian noises.

**Algorithm Used:** The PPL simulations were conducted using the *S-Plus* package (S-Plus90 [1]) implementation of PPL, where 3 and 5 hidden neurons were tried (with 5 and 7 maximum working hidden neurons used separately to avoid the overfitting). The *S-Plus* implementation is based on the Friedman code (Friedman85 [3]), which uses a Gauss-Newton method for updating the lower layer weights. To obtain a fair comparison, the BPL was implemented using a batch Gauss-Newton method (rather than the usual gradient descent, which is slower) on two-layer perceptrons with linear output neurons and nonlinear sigmoidal hidden neurons (Hwang90 [4], Hwang91 [5]), where 5 and 10 hidden neurons were tried.

**Independent Test Data Set:** The assessment of performance was done by comparing the fitted models with the "true" function counterparts on a large independent test set. Throughout all the simulations, we used the same set of test data for performance assessment, i.e., $\{g^{(j)}(x_{l1}, x_{l2})\}$, of size $N = 10000$, namely a regularly spaced grid on $[0, 1]^2$, defined by its marginals.

## 4.2   SIMULATION RESULTS IN LEARNING EFFICIENCY

To summarize the simulation results in learning efficiency, we focused on the chosen three aspects: *accuracy*, *parsimony*, and *speed*.

**Learning Accuracy:** The accuracy determined by the absolute $L_2$ error measure of the independent test data in both learning methods are quite comparable either trained by noiseless or noisy data (Hwang91 [5]). Note that our comparisons are based on 5 & 10 hidden neurons of BPLs and 3 & 5 hidden neurons of PPLs. The reason of choosing different number of hidden neurons will be explained in the learning parsimony section.

**Learning Parsimony:** In comparison with BPL, the PPL is more parsimonious in training all types of nonlinear functions, i.e., in order to achieve comparable accuracy to the BPLs for a two-layer perceptrons, the PPLs require fewer hidden neurons (more parsimonious) to approximate the desired true function (Hwang91 [5]). Several factors may contribute to this favorable performance. First and foremost, the data-smoothing technique creates more pertinent nonlinear nodal functions, so the network adapts more efficiently to the observation data without using too many terms (hidden neurons) of interpolative projections. Secondly, the batch Gauss-Newton BPL updates all the weights in the network simultaneously while the PPL updates cyclically (neuron-by-neuron and layer-by-layer), which allows the most recent updating information to be used in the subsequent updating. That is, more important projection directions can be determined first so that the less important projections can have a easier search (the same argument used in favoring the Gauss-Seidel method over the Jacobi method in an iterative linear equation solver).

**Learning Speed:** As we reported earlier (Maechler90 [7]), the PPL took much less time (1-2 order of magnitude speedup) in achieving accuracy comparable with that of the sequential gradient descent BPL. Interestingly, when compared with the batch Gauss-Newton BPL, the PPL took quite similar amount of time over all the simulations (under the same number of hidden neurons and the same convergence

threshold $\xi = 0.005$). In all simulations, both the BPLs and PPLs can converge under 100 iterations most of the time.

## 5   SENSITIVITY TO OUTLIERS

Both BPL's and PPL's are types of nonlinear least squares estimators. Hence like all least squares procedures, they are all sensitive to outliers. The outliers may come from large errors in measurements, generated by heavy tailed deviations from a Gaussian distribution for the noise $\epsilon_{li}$ in Eq. (1).

When in presence of additive Gaussian noises without outliers, most functions can be well approximated by 5-10 hidden neurons using BPL or with 3-5 hidden neurons using PPL. When the Gaussian noise is altered by adding one outlier, the BPL with 5-10 hidden neurons can still approximate the desired function reasonably well in general at the sacrifice of the magnified error around the vicinity of the outlier. If the number of outliers increases to 3 in the same corner, the BPL can only get a "distorted" approximation of the desired function. On the other hand, the PPL with 5 hidden neurons can successfully approximate the desired function and remove the single outlier. In case of three outliers, the PPL using simple data smoothing techniques can no longer keep its robustness in accuracy of approximation.

### Acknowledgements

This research was partially supported through grants from the National Science Foundation under Grant No. ECS-9014243.

## References

[1] *S-Plus* Users Manual (Version 3.0). Statistical Science Inc., Seattle, WA, 1990.

[2] D.L. Donoho and I.M. Johnstone. Projection–based approximation and a duality with kernel methods. The Annals of Statistics, Vol. 17, No. 1, pp. 58–106, 1989.

[3] J.H. Friedman. Classification and multiple regression through projection pursuit. Technical Report No. 12, Department of Statistics, Stanford University, January 1985.

[4] J. N. Hwang and P. S. Lewis. From nonlinear optimization to neural network learning. In Proc. *24th Asilomar Conf. on Signals, Systems, & Computers*, pp. 985-989, Pacific Grove, CA, November 1990.

[5] J. N. Hwang, H. Li, D. Martin, J. Schimert. The learning parsimony of projection pursuit and back-propagation networks. In *25th Asilomar Conf. on Signals, Systems, & Computers*, Pacific Grove, CA, November 1991.

[6] L.K. Jones. On a conjecture of Huber concerning the convergence of projection pursuit regression. The Annals of Statistics, Vol. 15, No. 2,880–882, 1987.

[7] M. Maechler, D. Martin, J. Schimert, M. Csoppenszky and J. N. Hwang. Projection pursuit learning networks for regression. in Proc. *2nd Int'l Conf. Tools for AI*, pp. 350-358, Washington D.C., November 1990.
